# Selectivity and Metaplasticity in a Unified Calcium-Dependent Model

**Luk Chong Yeung**
Physics Department and
Institute for Brain & Neural Systems
Brown University
Providence, RI 02912
yeung@physics.brown.edu

**Brian S. Blais**
Department of Science & Technology
Bryant College
Smithfield, RI 02917
Institute for Brain & Neural Systems
Brown University
bblais@bryant.edu

**Leon N Cooper**
Institute for Brain & Neural Systems
Physics Department and
Department of Neuroscience
Brown University
Providence, RI 02912
Leon_Cooper@brown.edu

**Harel Z. Shouval**
Institute for Brain & Neural Systems
and Physics Department
Brown University
Providence, RI 02912
Harel_Shouval@brown.edu

## Abstract

A unified, biophysically motivated Calcium-Dependent Learning model has been shown to account for various rate-based and spike time-dependent paradigms for inducing synaptic plasticity. Here, we investigate the properties of this model for a multi-synapse neuron that receives inputs with different spike-train statistics. In addition, we present a physiological form of metaplasticity, an activity-driven regulation mechanism, that is essential for the robustness of the model. A neuron thus implemented develops stable and selective receptive fields, given various input statistics

## 1 Introduction

Calcium influx through NMDA receptors is essential for the induction of diverse forms of bidirectional synaptic plasticity, such as rate-based [1, 2] and spike time-dependent plasticity (STDP) [3, 4]. Activation of NMDA receptors is also essential for functional plasticity in vivo [5]. An influential hypothesis holds that modest elevations of Ca above the basal line would induce LTD, while higher elevations would induce LTP[6, 7]. Based on these observations, a Unified Calcium Learning Model (UCM) has been proposed by Shouval *et al.* [8]. In this model, cellular activity is translated locally into the dendritic calcium concentrations $Ca_i$, through the voltage and time-dependence of the NMDA channels. The level of $Ca_i$ determines the sign and magnitude of synaptic plasticity as determined through a function of local calcium $\Omega(Ca_i)$(see Methods). A further assumption is that the Back-Propagating

Action Potentials (BPAP) has a slow after-depolarizing tail.

Implementation of this simple yet biophysical model has shown that it is sufficient to account for the outcome of different induction protocols of synaptic plasticity in a one-dimensional input space, as illustrated in Figure 1. In the pairing protocol, LTD occurs when LFS is paired with a small depolarization of the postsynaptic voltage while a larger depolarization yields LTP (Figure 1a), due to the voltage-dependence of the NMDA currents. In the rate-based protocol, low-frequency stimulation (LFS) gives rise to LTD while high-frequency stimulation (HFS) produces LTP (Figure 1b), due to the time-integration dynamics of the calcium transients. Finally, STDP gives LTD if a post-spike comes before a pre-spike within a time-window, and LTP if a post-spike comes after a pre-spike (Figure 1c); this is due to the coincidence-detector property of the NMDA receptors and the shape of the BPAP. In addition to these results, the model also predicts a previously uncharacterized pre-before-post depressing regime and rate-dependence of the STDP curve. These findings have had preliminary experimental support [9, 3, 10], and as will be shown have consequences in the multi-dimensional environment that impact the results of this work.

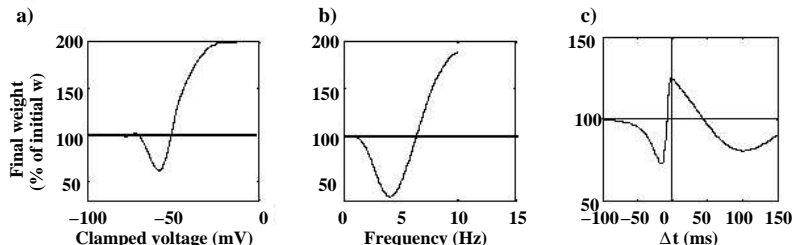

Figure 1: Calcium-Dependent Learning Rule and the various experimental plasticity-induction paradigms: implementation of (a) Pairing Protocol, (b) Rate-Dependent Plasticity and (c) Spike-Time Dependent Plasticity. The Pairing Protocol was simulated with a fixed input rate of 3 Hz; STDP curve is shown for 1 Hz. Notice the new pre-before-post depression regime.

In this study we investigate characteristics of the Calcium Control Hypothesis such as cooperativity and competition, and examine how they give rise to input selectivity. A neuron is called *selective* to a specific input pattern if it responds strongly to it and not to other patterns, which is equivalent to having a potentiated pathway to this pattern. Input selectivity is a general feature of neurons and underlies the formation of receptive fields and topographic mappings. We demonstrate that using the UCM alone, selectivity can arise, but only within a narrow range of parameters.

Metaplasticity, the activity-dependent modulation of synaptic plasticity, is essential for robustness of the BCM model [11]. Furthermore, it has significant experimental support [12]. Here we propose a more biologically realistic implementation, compatible with the Calcium Control Hypothesis, which is based on experimental observations [13]. We find that it makes the UCM model more robust significantly expanding the range of parameters that result in selectivity.

## 2 Selectivity to Spike Train Correlations

The development of neuronal selectivity, given any learning rule, depends on the statistical structures of the input environment. For spiking neurons, this structure

may include temporal, in addition to spatial statistics. One method of examining this feature is to generate input spike trains with different statistics across synapses. We use a simple scenario in which half of the synapses (group $B$) receive noisy Poisson spike trains with a mean rate $\langle r_{in} \rangle$, and the other half (group $A$), receive correlated spikes with the same rate $\langle r_{in} \rangle$. Input spikes in group A have an enhanced probability of arriving together (see Methods). One might expect that, by firing together, group $A$ will gain control of the post-synaptic firing times and thus be potentiated, while group $B$ will be depressed, in a manner similar to the STDP described by Song *et al.* [14]. In addition to the 100 excitatory neurons our neuron receives 20 inhibitory inputs.

The results are shown in Figure 2. There exists a range of input frequencies (Figure 2a, *left*) at which segregation occurs between the correlated and uncorrelated groups. The cooperativity among the synapses in group $A$ enhances its probability of generating a post-spike, which, through the BPAP causes strong depolarization. Since the NMDA channels are still open due to a recent pre-spike, this is likely to potentiates these synapses in a Hebbian-associative fashion. Group $B$ will fire with equal probability before and after a post-spike which, given a sufficiently low NMDA receptor conductance, ensures that, on average, depression takes place. At the final state, the output spike train is irregular (Figure 2a, *right*) but its rate is stable (Figure 2a, *center*), indicating that the system had reached a fixed point with a balance between excitation and inhibition.

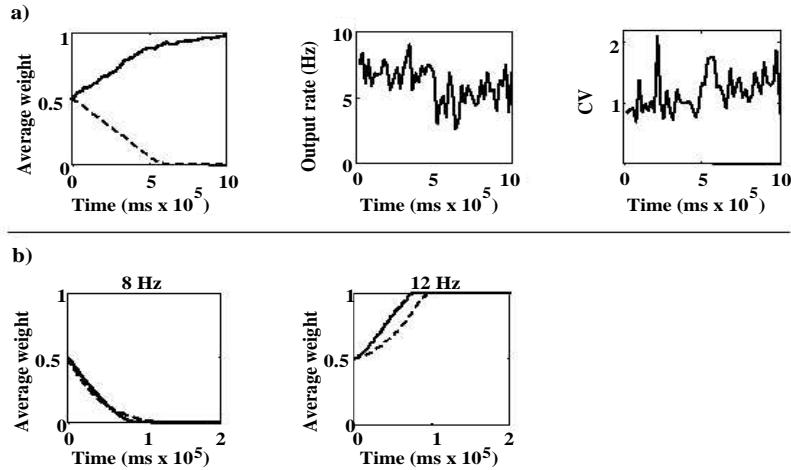

Figure 2: Segregation of the synapses for different input structures. (a) Segregation at 10 Hz. *Left*, time evolutions of the average synaptic weight for the groups $A$ (solid) and $B$ (dashed). *Center*, the output rate, calculated as the number of output spikes over non-overlapping time bins of 20 seconds. *Right*, the coefficient of variation, CV = std($isi$)/ mean($isi$), where $isi$ is the interspike interval of the output train. (b) Results for 8 Hz (*left*) and 12 Hz (*right*). All the synapses are potentiated and depressed, respectively.

These results, however, are sensitive to the simulation parameters. In fact, a slight change in the value of $\langle r_{in} \rangle$ disrupts the segregation described previously (Figure 2b). For too high or too low values of $\langle r_{in} \rangle$, both channels are potentiated and depressed, respectively. This occurs because, unlike standard STDP models, the unified model exhibits frequency dependence in addition to spike-time dependence. This suggests that a stabilizing mechanism must be incorporated into the model.

# 3   Metaplasticity

In the BCM theory the threshold between LTD and LTP moves as a function of the history of postsynaptic activity [11]. This type of activity-dependent regulation of the properties of synaptic plasticity, or *metaplasticity*, was developed to ensure selectivity and stability. Experimental results have linked some forms of metaplasticity to the magnitude of the NMDA conductance; it is shown that as the cellular activity increases, NMDA conductance is down-regulated, and vice-versa [15, 16, 13, 17]. Under the Calcium Control Hypothesis, this sets the ground for a more physiological formulation of metaplasticity [18].

NMDA conductance is interpreted here as the total number $(g_m)$ of NMDA channels inserted in the membrane of the postsynaptic terminal. Consider a simple kinetic model in which additional channels can be inserted from an intracellular pool $(g_i)$ or removed and returned to the pool in an activity dependent manner. We assume a fixed removal rate $k_-$ and a voltage sensitive insertion rate $k_+V^\alpha$:

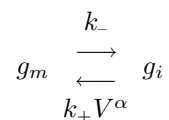

(Our results are not very sensitive to the details of the voltage dependence of insertion and removal rates)

This scheme leads us to a dynamic equation for $g_m$, $\dot{g}_m = -\left(k_- + k_+V^\alpha\right)g_m + k_+V^\alpha g_t$, where $g_t$ is a normalizing factor, $g_t = g_m + g_i$. The fixed point is:

$$g_m^* = \frac{g_t}{k_-/(k_+V^\alpha) + 1} \tag{1}$$

If, in this model, cellular activity is translated into Ca, then $g_m$ can be loosely interpreted as the inverse of the BCM sliding threshold $\theta_m$ [18]. Notice that in the original form of BCM, $\theta_m$ is the time average of a non-linear function of the postsynaptic the activity level. In order to achieve competition, $g_m$ should not depend solely on local (synaptic) variables, but should rather detect changes of the global patterns of cellular activity. Here, the activity-signaling global variable is taken to be postsynaptic membrane potential.

Implementation of metaplasticity widens significantly the range of input frequencies for which segregation between the weights of correlated and uncorrelated synapses is observed; this is shown in Figure 3a. At low spiking activity, the subthreshold depolarization levels prevent significant inward Ca currents. Under these conditions metaplasticity causes $g_m$ to grow. Persistent post-spike generation will lead $g_m$ and therefore Ca to decrease, hence scaling the synaptic weights downwards. Competition arises as the system searches for the balance between the selective positive feed-back of a standard Hebbian rule and the overall negative feed-back of a sliding threshold mechanism. However, consistent with the rate-based protocol described before, at too low and too high $\langle r_{in} \rangle$ selectivity is disrupted, and the synapses will eventually all depress or potentiate, regardless of the statistical structures of the stimulus. Strengthening the correlation increases segregation (Figure 3b), demonstrating the effects of lateral cooperativity in potentiation. On the other hand, increasing the fraction of correlated inputs weakens the final weight of the correlated group (Figure 3c), suggesting that less potentiation is needed to control the output spike-timing. Notice that in the presence of metaplasticity, no upper saturation limit is required; the equilibrium of the fixed point is homeostatic, rather than imposed.

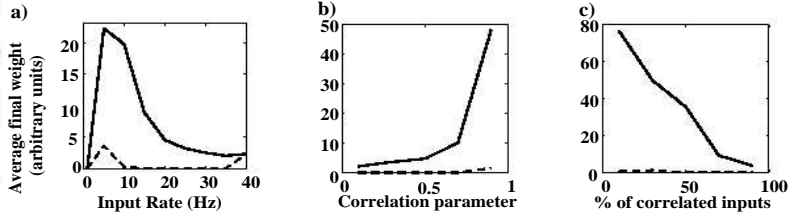

Figure 3: The effects of metaplasticity. (a) The weights segregate within the range of input frequency = [5, 35] Hz in a half correlated (solid), half uncorrelated (dashed) input environment; shown are the average final weights within each group, correlation parameter $c = 0.8$ (see Methods). (b) The average final weight as a function of the correlation parameter, $\langle r_{in} \rangle = 10$ Hz. (c) The average final weight as a function of the fraction of correlated inputs, $\langle r_{in} \rangle = 10$ Hz, $c = 0.8$.

## 4  Selectivity to patterns of rate distribution

An alternative input environment is one in which the rates vary across the synapses and over time. This is a plausible representation for sensory neurons that are differentially excited. A straightforward method is to use rate distributions that are piecewise constant. We use a simple example in which the rate distributions are non-overlapping square patterns, as illustrated in Figure 4a (see Methods). The patterns are randomly presented to the neuron, being switched at regular epochs. Since the mean switching time is constant and much smaller than the time constant of learning, each synapse receives the same average input over time. However, we observe that, after training, the neuron spontaneously breaks the symmetry, as a subset of synapses becomes potentiated, while others are depressed (Figure 4b). It should be noticed that, because the choice of the training pattern at each epoch is random, the selected pattern is different at each run. Due to metaplasticity, these results are robust across different pattern amplitudes and pattern dimensions (not shown).

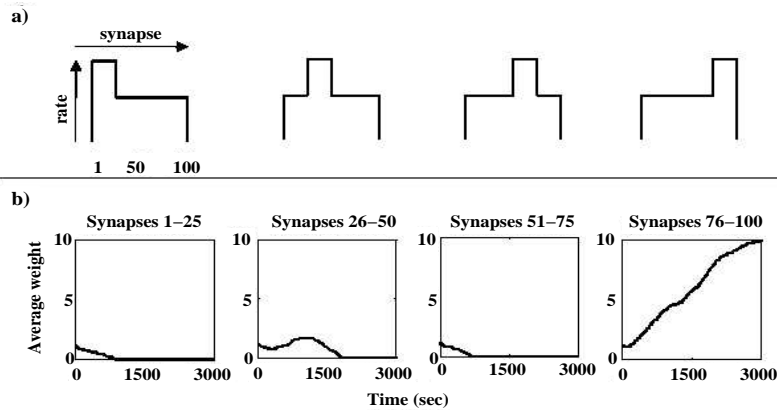

Figure 4: (a) Four non-overlapping patterns of input rate distribution and (b) the average weight evolution of each channel. In this particular simulation, the higher and the lower rates correspond to 30 Hz and 10 Hz, respectively. The final state of the neuron is one that is selective to the last pattern ( a), *left most*).

# 5 Discussion

Neurons in many cortical areas develop receptive fields that are selective to a small subset of stimulating inputs. This property has been shown to be experience-dependent [19, 20] and also dependent on NMDA receptors[5, 21]. It is likely, therefore, that receptive field formation relies on the same type of NMDA-dependent synaptic plasticity observed *in vitro* [1, 2, 4]. Previous work has shown that these *in vitro* rate and spike time-induced plasticity can be accounted for by the biologically-inspired Unified Calcium Model. In this work, we have shown that the same model can lead to the experience-dependent development of neuronal selectivity.

Metaplasticity adds robustness to the system and reinforces temporal competition between input patterns [11] , by controlled scaling of NMDAR currents. We have shown here that even in simple input environments there is segregation among the synaptic strengths, depending on the temporal input statistics of different channels. This is analogous to the explanation of ocular dominance that depends on temporal competition [22], and is likely to hold with more realistic assumptions.

Because the UCM is responsive to input rates, in addition to spike-timing, we are able to achieve selectivity for rate-distribution patterns in spiking neurons that is comparable to the selectivity obtained in simplified, continuous-valued systems [23]. This result suggests that the coexistence and complementarity of rate- and spike time-dependent plasticities, previously demonstrated for a one-dimensional neuron [8], can also be extended to multi-dimensional input environments. We are currently investigating the formation of receptive fields in more realistic environments, such as natural stimuli and examining how the their statistical properties can be translated into a physiological mechanism for emergence of input selectivity.

# 6 Methods

We simulate a single neuron with 20 non-plastic inhibitory synapses and 100 excitatory synapses undergoing the Calcium-Dependent learning rule:

$$\dot{w}_i = \eta(\mathrm{Ca}_i)\left(\Omega(\mathrm{Ca}_i) - \lambda w\right),\qquad(2)$$

where $w_i$ is the synaptic weight of the synapse $i$, $i = 1, ..., 100$, $\eta$ is a linear calcium-dependent learning rate $\eta = 10^{-3}\mathrm{Ca}$ and $\Omega$ is a difference of sigmoids: $\Omega = \sigma(\mathrm{Ca}, \alpha_1, \beta_1) - 0.5\sigma(\mathrm{Ca}, \alpha_2, \beta_2)$, with $\sigma(x, a, b) := \exp(b(x-a))\left[1 + \exp(b(x - a))\right]^{-1}$ and $(\alpha_1, \beta_1, \alpha_2, \beta_2) = (0.25, 60, 0.4, 20)$. Here, we use $\lambda = 0$. The initial condition for all weights is 0.5; additionally, $w_i$ is constrained within hard boundaries: $w_i \in [0, 1]$ for the cases where no metaplasticity is used.

The NMDA-mediated calcium concentration varies as:

$$\frac{d\mathrm{Ca}_i}{dt} = I - \frac{\mathrm{Ca}_i}{\tau_{\mathrm{Ca}}},\qquad(3)$$

where $I$ is the NMDA current and $\tau_{\mathrm{Ca}} = 20$ ms is the passive decay time constant [24]. $I$ depends on the association between pre-spike times and postsynaptic depolarization level, described by $I = g_m f(t, t_{pre})\mathcal{H}(V)$ [7]. At the non-metaplastic cases, we use $g_m = 2.53 \times 10^{-4} \mu\mathrm{M}/(\mathrm{mV.ms})$. Upon a pre-spike, $f$ reaches its peak value of 1. 70% of this value decays with time constant $\tau_f^{\mathrm{N}} = 50$ ms, the remaining decays with time constant $\tau_s^{\mathrm{N}} = 200$ ms. $\mathcal{H}$ is the magnesium-block function:

$$\mathcal{H}(V) = \frac{(V - V_{rev})}{1 + e^{-0.062V}/3.57},\qquad(4)$$

with the reversal potential for calcium $V_{rev} = 130$ mV.

The dynamics of the membrane potential is simulated with the standard Integrate-and-Fire model:

$$\frac{dV_m(t)}{dt} = \frac{1}{\tau_m} V_{rest} - V_m(t) + G_{ex}(t)\left(V_{ex} - V_m(t)\right) + G_{in}(t)\left(V_{in} - V_m(t)\right), \quad (5)$$

where $\tau_m = 20$ ms, $(V_{rest}, V_{ex}, V_{in}) = (-65, 0, 65)$ mV. If a pre-spike arrives at the excitatory [inhibitory] synapse $i$, $G_{ex[in]}(t) = G_{ex[in]}(t-1) + g_{ex[in]}^{max} g_i$; otherwise, $G_{ex}$ and $G_{in}$ decay exponentially with time constant $\tau = 5$ ms. For excitatory and inhibitory synapses, $(g_i, g^{max}) = (w_i, 0.09)$ and $(1, 0.3)$ respectively. If $V_m(t)$ reaches firing threshold of -55 mV, a post-spike is generated and the BPAP is updated to its peak value of 60 mV. 75% of this value decays rapidly ($\tau_f^{B} = 3$ ms) and the remaining decays slowly ($\tau_s^{B} = 35$ ms) [25]. The voltage at the synaptic site is thus given by the sum $V = V_m + \text{BPAP}$.

To implement input correlations, we adopt the method used by [26]. Let the number of correlated input be $N$. For a pre-assigned correlation parameter $c$, $N_0$ Poisson events are generated, $N_0 = N + \sqrt{c}(1 - N)$, and, at each time step, randomly distributed among the $N$ synapses. It is clear that each resulting spike-train still has the same Poisson distribution, but with a probability of spiking together with other synapses.

For simulations involving different rates, the 100 synapses were first divided into 4 channels of 25 synapses. Time epochs were generated according to an exponential distribution of mean $\tau_c = 500$ ms. At each epoch, one of the channels was randomly chosen and assigned a mean rate $r^*$, while others receive spike-trains with mean rate $r < r^*$.

For metaplasticity in Equation 1, we use the parameters: $k_-/(k_+) = 9.1739 \times 10^7$, $g_t = -0.0184$ and $\alpha = 4$. All of the simulations use time steps of $dt = 1$ ms.

**Acknowledgments**

This work is partly funded by the Brown Brain Science Program Burroughs-Wellcome Fund fellowship program. The authors thank the members of the Institute for Brain and Neural Systems and the participants of the 2001 EU Summer School on Computational Neuroscience for helpful conversations.

## References

[1] T.V.P. Bliss and G.L. Collingridge. A synaptic model of memory; long-term potentiation the hippocampus. *Nature*, 361:31–9, 1993.

[2] S.M. Dudek and M.F. Bear. Homosynaptic long-term depression in area CA1 of hippocampus and the effects on NMDA receptor blockade. *Proc. Natl. Acad. Sci.*, 89:4363–7, 1992.

[3] H. Markram, J. Lübke, M. Frotscher, and B. Sakmann. Regulation of synaptic efficacy by coincidence of postsynaptic APs and EPSPs. *Science*, 275:213–5, 1997.

[4] G. Bi and M. Poo. Synaptic modifications in cultured hippocampal neurons: Dependence on spike timing, synaptic strength, and postsynaptic cell type. *J. Neurosci.*, 18 (24):10464–72, 1998.

[5] A. Kleinschmidt, M.F. Bear, and W. Singer. Blockade of NMDA receptors disrupts experience-dependent plasticity of kitten striate cortex. *Science*, 238:355–358, 1987.

[6] M.F. Bear, L.N Cooper, and F.F. Ebner. A physiological basis for a theory of synapse modification. *Science*, 237:42–8, 1987.

[7] J.A. Lisman. A mechanism for the Hebb and the anti-Hebb processes underlying learning and memory. *Proc. Natl. Acad. Sci.*, 86:9574–8, 1989.

[8] H.Z. Shouval, M.F. Bear, and L.N Cooper. A unified theory of nmda receptor-dependent bidirectional synaptic plasticity. *Proc. Natl. Acad. Sci.*, 99:10831–6, 2002.

[9] M. Nishiyama, K. Hong, K. Mikoshiba, M.M. Poo, and K. Kato. Calcium stores regulate the polarity and input specificity of synaptic modification. *Nature*, 408:584–8, 2000.

[10] P.J. Sjöström, G.G. Turrigiano, and S.B. Nelson. Rate, timing, and cooperativity jointly determine cortical synaptic plasticity. *Neuron*, 32:1149–64, 2001.

[11] E.L. Bienenstock, L.N Cooper, and P.W. Munro. Theory for the development of neuron selectivity: orientation specificity and binocular interaction in visual cortex. *J. Neurosci.*, 2:32–48, 1982.

[12] A. Kirkwood, M.G. Rioult, and M.F. Bear. Experience-dependent modification of synaptic plasticity in visual cortex. *Nature*, 381:526–8, 1996.

[13] B.D. Philpot, A.K. Sekhar, H.Z. Shouval, and M.F. Bear. Visual experience and deprivation bidirectionally modify the composition and function of NMDA receptors in visual cortex. *Neuron*, 29:157–69, 2001.

[14] S. Song, K.D. Miller, and L.F. Abbott. Competitive hebbian learning through spike-timing dependent synaptic plasticity. *Nature Neurosci.*, 3:919–26, 2000.

[15] G. Carmignoto and S. Vicini. Activity dependent increase in NMDA receptor responses during development of visual cotex. *Science*, 258:1007–11, 1992.

[16] E.M. Quinlan, B.D. Philpot, R.L. Huganir, and M.F. Bear. Rapid, experience-dependent expression of synaptic NMDA receptors in visual cortex in vivo. *Nature Neurosci.*, 2(4):352–7, 1999.

[17] A.J. Watt, M.C.W. van Rossum, K.M. MacLeod, S.B. Nelson, and G.G. Turrigiano. Activity co-regulates quantal ampa and nmda currents at neocortical synapses. *Neuron*, 26:659–70, 2000.

[18] H.Z. Shouval, G.C. Castellani, L.C. Yeung, B.S. Blais, and L.N Cooper. Converging evidence for a simplified biophysical model of synaptic plasticity. *Bio. Cyb.*, 87:383–91, 2002.

[19] Y. Frégnac and M. Imbert. Early development of visual cortical cells in normal and dark reared kittens: relationship between orientation selectivity and ocular dominance. *J. Physiol. Lond.*, 278:27–44, 1978.

[20] B. Chapman, M.P. Stryker, and T. Bonhoeffer. Development of orientation preference maps in ferret primary visual cortex. *J. Neurosci.*, 16:6443–53, 1996.

[21] A.S. Ramoa, A.F. Mower, D. Liao, and S.I. Jafri. Suppression of cortical nmda receptor function prevents development of orientation selectivity in the primary visual cortex. *J. Neurosci.*, 21:4299–309, 2001.

[22] B.S. Blais, H.Z. Shouval, and L.N Cooper. The role of presynaptic activity in monocular deprivation: Comparison of homosynaptic and heterosynaptic mechanisms. *Proc. Natl. Acad. Sci.*, 96:1083–7, 1999.

[23] E.E. Clothiaux, L.N Cooper, and M.F. Bear. Synaptic plasticity in visual cortex: Comparison of theory with experiment. *J. Neurophys.*, 66:1785–804, 1991.

[24] B.L. Sabatini, T.G. Oerthner, and K. Svoboda. The life cycle of ca2+ ions in dendritic spines. *Neuron*, 33:439–52, 2002.

[25] J.C. Magee and D. Johnston. A synaptically controlled, associative signal for hebbian plasticity in hippocampal neurons. *Science*, 275:209–13, 1997.

[26] M. Rudolph and A. Destexhe. Correlation detection and resonance in neural systems with distributed noise sources. *Phys. Rev. Lett.*, 86(16):3662–5, 2001.
